# Hierarchical Multitask Structured Output Learning for Large-Scale Sequence Segmentation

**Nico Görnitz**[1]
Technical University Berlin,
Franklinstr. 28/29, 10587 Berlin, Germany
Nico.Goernitz@tu-berlin.de

**Christian Widmer**[1]
FML of the Max Planck Society
Spemannstr. 39, 72070 Tübingen, Germany
Christian.Widmer@tue.mpg.de

**Georg Zeller**
European Molecular Biology Laboratory
Meyerhofstr. 1, 69117 Heidelberg, Germany
Georg.Zeller@gmail.com

**André Kahles**
FML of the Max Planck Society
Spemannstr. 39, 72070 Tübingen, Germany
Andre.Kahles@tue.mpg.de

**Sören Sonnenburg**[2]
TomTom
An den Treptowers 1, 12435 Berlin, Germany
Soeren.Sonnenburg@tomtom.com

**Gunnar Rätsch**
FML of the Max Planck Society
Spemannstr. 39, 72070 Tübingen, Germany
Gunnar.Raetsch@tue.mpg.de

## Abstract

We present a novel regularization-based Multitask Learning (MTL) formulation for Structured Output (SO) prediction for the case of hierarchical task relations. Structured output prediction often leads to difficult inference problems and hence requires large amounts of training data to obtain accurate models. We propose to use MTL to exploit additional information from related learning tasks by means of hierarchical regularization. Training SO models on the combined set of examples from multiple tasks can easily become infeasible for real world applications. To be able to solve the optimization problems underlying multitask structured output learning, we propose an efficient algorithm based on bundle-methods. We demonstrate the performance of our approach in applications from the domain of computational biology addressing the key problem of gene finding. We show that 1) our proposed solver achieves much faster convergence than previous methods and 2) that the Hierarchical SO-MTL approach outperforms considered non-MTL methods.

## 1   Introduction

In Machine Learning, model quality is most often limited by the lack of sufficient training data. When data from different, but related tasks, is available, it is possible to exploit it to boost the performance of each task by transferring relevant information. Multitask learning (MTL) considers the problem of inferring models for several tasks simultaneously, while imposing regularity criteria or shared representations in order to allow learning across tasks. This has been an active research focus and various methods (e.g., [5, 8]) have been explored, providing empirical findings [16] and theoretical foundations [3, 4]. Recently, also the relationships between tasks have been studied (e.g., [1]) assuming a cluster relationship [11] or a hierarchy [6, 23, 13] between tasks. Our proposed method follows this line of research in that it exploits externally provided hierarchical task relations. The generality of regularization-based MTL approaches makes it possible to extend them beyond the simple cases of classification or regression to Structured Output (SO) learning problems

[14, 2, 21, 10]. Here, the output is not in the form of a discrete class label or a real valued number, but a structured entity such as a label sequence, a tree, or a graph. One of the main contributions of this paper is to explicitly extend a regularization-based MTL formulation to the SVM-struct formulation for SO prediction [2, 21]. SO learning methods can be computationally demanding, and combining information from several tasks leads to even larger problems, which renders many interesting applications infeasible. Hence, our second main contribution is to provide an efficient solver for SO problems which is based on bundle methods [18, 19, 7]. It achieves much faster convergence and is therefore an essential tool to cope with the demands of the MTL setting.

SO learning has been successfully applied in the analysis of images, natural language, and sequences. The latter is of particular interest in computational biology for the analysis of DNA, RNA or protein sequences. This field moreover constitutes an excellent application area for MTL [12, 22]. In computational biology, one often uses supervised learning methods to model biological processes in order to predict their outcomes and ultimately understand them better. Due to the complexity of many biological mechanisms, rich computational models have to be developed, which in turn require a reasonable amount of training data. However, especially in the biomedical domain, obtaining labeled training examples through experiments can be costly. Thus, combining information from several related tasks can be a cost-effective approach to best exploit the available label data. When transferring label information across tasks, it often makes sense to assume hierarchical task relations. In particular, in computational biology, where evolutionary processes often impose a task hierarchy [22]. For instance, we might be interested in modeling a common biological mechanism in several organisms such that each task corresponds to one organism. In this setting, we expect that the longer the common evolutionary history between two organisms, the more beneficial it is to share information between the corresponding tasks. In this work, we chose a challenging problem from genome biology to demonstrate that our approach is practically feasible in terms of speed and accuracy. In *ab initio* gene finding [17], the task is to build an accurate model of a gene and subsequently use it to predict the gene content of newly sequenced genomes or to refine existing annotations. Despite many commonalities between sequence features of genes across organisms, sequence differences have made it very difficult to build universal gene finders that achieve high accuracy in cross-organism prediction. This problem is hence ideally suited for the application of the proposed SO-MTL approach.

## 2   Methods

Regularization based supervised learning methods, such as the SVM or Logistic Regression play a central role in many applications. In its most general form, such a method consists of a loss function $L$ that captures the error with respect to the training data $S = \{(\boldsymbol{x}_1, y_1), \ldots, (\boldsymbol{x}_n, y_n)\}$ and a regularizer $R$ that penalizes model complexity

$$J(\mathbf{w}) = \sum_{i=1}^{n} L(\mathbf{w}, \boldsymbol{x}_i, y_i) + R(\mathbf{w}).$$

In the case of Multitask Learning (MTL), one is interested in obtaining several models $\mathbf{w}_1, ..., \mathbf{w}_T$ based on $T$ associated sets of examples $S_t = \{(\boldsymbol{x}_1, y_1), \ldots, (\boldsymbol{x}_{n_t}, y_{n_t})\}$, $t = 1, \ldots, T$. To couple individual tasks, an additional regularization term $R_{MTL}$ is introduced that penalizes the disagreement between the individual models (e.g., [1, 8]):

$$J(\mathbf{w}_1, ..., \mathbf{w}_T) = \sum_{t=1}^{T} \left( \sum_{i=1}^{n_t} L(\mathbf{w}, \boldsymbol{x}_i, y_i) + R(\mathbf{w}_t) \right) + R_{MTL}(\mathbf{w}_1, ..., \mathbf{w}_T).$$

Special cases include $T = 2$ and $R_{MTL}(\mathbf{w}_1, \mathbf{w}_2) = \gamma \, ||\mathbf{w}_1 - \mathbf{w}_2||$ (e.g., [8, 16]), where $\gamma$ is a hyper-parameter controlling the strength of coupling of the solutions for both tasks. For more than two tasks, the number of coupling terms and hyper-parameters can rise quadratically leading to a difficult model-selection problem.

### 2.1   Hierarchical Multitask Learning (HMTL)

We consider the case where tasks correspond to leaves of a tree and are related by its inner nodes. In [22], the case of taxonomically organized two-class classification tasks was investigated, where each task corresponds to a species (taxon). The idea was to mimic biological evolution that is assumed to

generate more specialized molecular processes with each speciation event from root to leaf. This is implemented by training on examples available for nodes in the current subtree (i.e., the tasks below the current node), while similarity to the parent classifier is induced through regularization. Thus, for each node $n$, one solves the following optimization problem,

$$(\mathbf{w}_n^*, b_n^*) = \underset{\mathbf{w}, b}{\operatorname{argmin}} \left\{ \frac{1}{2} \left( (1 - \gamma) \|\mathbf{w}\|^2 + \gamma \|\mathbf{w} - \mathbf{w}_p^*\|^2 \right) + C \sum_{(\boldsymbol{x}, y) \in S} \ell \left( \langle \boldsymbol{x}, \mathbf{w} \rangle + b, y \right) \right\}, \quad (1)$$

where $p$ is the parent node of $n$ (with the special case of $\mathbf{w}_p^* = \mathbf{0}$ for the root node), $\ell$ is an appropriate loss function (e.g., the hinge-loss). The hyper-parameter $\gamma \in [0, 1]$ determines the contribution of regularization from the origin vs. the parent node's parameters (i.e., the strength of coupling between the node and its parent). The above problem can be equivalently rewritten as:

$$(\mathbf{w}_n^*, b_n^*) = \underset{\mathbf{w}, b}{\operatorname{argmin}} \left\{ \frac{1}{2} \|\mathbf{w}\|^2 - \gamma \langle \mathbf{w}, \mathbf{w}_p^* \rangle + C \sum_{(\boldsymbol{x}, y) \in S} \ell \left( \langle \boldsymbol{x}, \mathbf{w} \rangle + b, y \right) \right\}. \quad (2)$$

For $\gamma = 0$, the tasks completely decouple and can be learnt independently. The parameters for the root node correspond to the globally best model. We will refer to these two cases as base-line methods for comparisons in the experimental section.

## 2.2 Structured Output Learning and Extensions for HMTL

In contrast to binary classification, elements from the output space $\Upsilon$ (e.g., sequences, trees, or graphs) of structured output problems have an inherent structure which makes more sophisticated, problem-specific loss functions desirable. The loss between the true label $\boldsymbol{y} \in \Upsilon$ and the predicted label $\hat{\boldsymbol{y}} \in \Upsilon$ is measured by a loss function $\Delta : \Upsilon \times \Upsilon \to \Re^+$. A widely used approach to predict $\hat{\boldsymbol{y}} \in \Upsilon$ is the use of a linearly parametrized model given an input vector $\boldsymbol{x} \in \mathcal{X}$ and a joint feature map $\Psi : \mathcal{X} \times \Upsilon \to \mathcal{H}$ that captures the dependencies between input and output (e.g., [21]):

$$\hat{\boldsymbol{y}}_{\mathbf{w}}(\boldsymbol{x}) = \underset{\bar{\boldsymbol{y}} \in \Upsilon}{\operatorname{argmax}} \quad \langle \mathbf{w}, \Psi(\boldsymbol{x}, \bar{\boldsymbol{y}}) \rangle.$$

The most common approaches to estimate the model parameters $\mathbf{w}$ are based on structured output SVMs (e.g., [2, 21]) and conditional random fields (e.g., [14]; see also [10]). Here we follow the approach taken in [21, 15], where estimating the parameter vector $\mathbf{w}$ amounts to solving the following optimization problem

$$\underset{\mathbf{w} \in \mathcal{H}}{\min} \left\{ R(\mathbf{w}) + C \sum_{i=1}^{n} \ell(\underset{\bar{\boldsymbol{y}} \in \Upsilon}{\max} \langle \mathbf{w}, \Psi(\boldsymbol{x}_i, \bar{\boldsymbol{y}}) \rangle + \Delta(\boldsymbol{y}_i, \bar{\boldsymbol{y}}) - \langle \mathbf{w}, \Psi(\boldsymbol{x}_i, \boldsymbol{y}_i) \rangle) \right\}, \quad (3)$$

where $R(\mathbf{w})$ is a regularizer and $\ell$ is a loss function. For $\ell(a) = \max(0, a)$ and $R(\mathbf{w}) = \|\mathbf{w}\|_2^2$ we obtain the structured output support vector machine [21, 2] with margin rescaling and hinge-loss.

It turns out that we can combine the structured output formulation with hierarchical multitask learning in a straight-forward way. We extend the regularizer $R(\mathbf{w})$ in (3) with a $\gamma$-parametrized convex combination of a multitask regularizer $\frac{1}{2}\|\mathbf{w} - \mathbf{w}_p\|_2^2$ with the original term. When $R(\mathbf{w}) = \frac{1}{2}\|\mathbf{w}\|_2^2$ and omitting constant terms, we arrive at $R_{p,\gamma}(\mathbf{w}) = \frac{1}{2}\|\mathbf{w}\|_2^2 - \gamma \langle \mathbf{w}, \mathbf{w}_p \rangle$. Thus we can apply the described hierarchical multitask learning approach and solve for every node the following optimization problem:

$$\underset{\mathbf{w} \in \mathcal{H}}{\min} \left\{ R_{p,\gamma}(\mathbf{w}) + C \sum_{i=1}^{n} \ell(\underset{\bar{\boldsymbol{y}} \in \Upsilon}{\max} \langle \mathbf{w}, \Psi(\boldsymbol{x}_i, \bar{\boldsymbol{y}}) \rangle + \Delta(\boldsymbol{y}_i, \bar{\boldsymbol{y}}) - \langle \mathbf{w}, \Psi(\boldsymbol{x}_i, \boldsymbol{y}_i) \rangle) \right\} \quad (4)$$

A major difficulty remains: solving the resulting optimization problems which now can become considerably larger than for the single-task case.

## 2.3 A Bundle Method for Efficient Optimization

A common approach to obtain a solution to (3) is to use so-called cutting-plane or column-generation methods. Here one considers growing subsets of all possible structures and solves restricted optimization problems. An algorithm implementing a variant of this strategy based on primal optimization is given in the appendix (similar in [21]). Cutting-plane and column generation techniques

often converge slowly. Moreover, the size of the restricted optimization problems grows steadily and solving them becomes more expensive in each iteration. Simple gradient descent or second order methods can not be directly applied as alternatives, because (4) is continuous but non-smooth. Our approach is instead based on bundle methods for regularized risk minimization as proposed in [18, 19] and [7]. In case of SVMs, this further relates to the OCAS method introduced in [9]. In order to achieve fast convergence, we use a variant of these methods adapted to structured output learning that is suitable for hierarchical multitask learning.

We consider the objective function $J(\mathbf{w}) = R_{p,\gamma}(\mathbf{w}) + L(\mathbf{w})$, where

$$L(\mathbf{w}) := C \sum_{i=1}^{n} \ell(\max_{\bar{\boldsymbol{y}} \in \Upsilon} \{\langle \mathbf{w}, \Psi(\boldsymbol{x}_i, \bar{\boldsymbol{y}}) \rangle + \Delta(\boldsymbol{y}_i, \bar{\boldsymbol{y}})\} - \langle \mathbf{w}, \Psi(\boldsymbol{x}_i, \boldsymbol{y}_i) \rangle)$$

and $R_{p,\gamma}(\mathbf{w})$ is as defined in Section 2.2. Direct optimization of $J$ is very expensive as computing $L$ involves computing the maximum over the output space. Hence, we propose to optimize an estimate of the empirical loss $\hat{L}(\mathbf{w})$, which can be computed efficiently. We define the estimated empirical loss $\hat{L}(\mathbf{w})$ as

$$\hat{L}(\mathbf{w}) := C \sum_{i=1}^{N} \ell \left( \max_{(\Psi, \Delta) \in \Gamma_i} \{\langle \mathbf{w}, \Psi \rangle + \Delta\} - \langle \mathbf{w}, \Psi(\boldsymbol{x}_i, \boldsymbol{y}_i) \rangle \right).$$

Accordingly, we define the estimated objective function as $\hat{J}(\mathbf{w}) = R_{p,\gamma}(\mathbf{w}) + \hat{L}(\mathbf{w})$. It is easy to verify that $J(\mathbf{w}) \geq \hat{J}(\mathbf{w})$. $\Gamma_i$ is a set of pairs $(\Psi(\boldsymbol{x}_i, \boldsymbol{y}), \Delta(\boldsymbol{y}_i, \boldsymbol{y}))$ defined by a suitably chosen, growing subset of $\Upsilon$, such that $\hat{L}(\mathbf{w}) \to L(\mathbf{w})$ (cf. Algorithm 1).

In general, bundle methods are extensions of cutting plane methods that use a prox-function to stabilize the solution of the approximated function. In the framework of regularized risk minimization, a natural prox-function is given by the regularizer. We apply this approach to the objective $\hat{J}(\mathbf{w})$ and solve

$$\min_{\mathbf{w}} R_{p,\gamma}(\mathbf{w}) + \max_{i \in I} \{\langle \boldsymbol{a}_i, \mathbf{w} \rangle + b_i\} \tag{5}$$

where the set of cutting planes $\boldsymbol{a}_i$, $b_i$ lower bound $\hat{L}$. As proposed in [7, 19], we use a set $I$ of limited size. Moreover, we calculate an aggregation cutting plane $\bar{\boldsymbol{a}}$, $\bar{b}$ that lower bounds the estimated empirical loss $\hat{L}$. To be able to solve the primal optimization problem in (5) in the dual space as proposed by [7, 19], we adopt an elegant strategy described in [7] to obtain the aggregated cutting plane $(\bar{\boldsymbol{a}}', \bar{b}')$ using the dual solution $\boldsymbol{\alpha}$ of (5):

$$\bar{\boldsymbol{a}}' = \sum_{i \in I} \alpha_j \boldsymbol{a}_i \qquad \text{and} \qquad \bar{b}' = \sum_{i \in I} \alpha_i b_i. \tag{6}$$

The following two formulations reach the same minimum when optimized with respect to $\mathbf{w}$:

$$\min_{\mathbf{w} \in \mathcal{H}} \left\{ R_p(\mathbf{w}) + \max_{i \in I} \langle \boldsymbol{a}_i, \mathbf{w} \rangle + b_i \right\} = \min_{\mathbf{w} \in \mathcal{H}} \{R_p(\mathbf{w}) + \langle \bar{\boldsymbol{a}}', \mathbf{w} \rangle + \bar{b}'\}.$$

This new aggregated plane can be used as an additional cutting plane in the next iteration step. We therefore have a monotonically increasing lower bound on the estimated empirical loss and can remove previously generated cutting planes without compromising convergence (see [7] for details).

The algorithm is able to handle any (non-)smooth convex loss function $\ell$, since only the subgradient needs to be computed. This can be done efficiently for the hinge-loss, squared hinge-loss, Huber-loss, and logistic-loss.

The resulting optimization algorithm is outlined in Algorithm 1. There are several improvements possible: For instance, one can bypass updating the empirical risk estimates in line 6, when $L(\mathbf{w}^{(k)}) - \hat{L}(\mathbf{w}^{(k)}) \leq \epsilon$. Finally, while Algorithm 1 was formulated in primal space, it is easy to reformulate in dual variables making it independent of the dimensionality of $\mathbf{w} \in \mathcal{H}$.

## 2.4 Taxonomically Constrained Model Selection

Model selection for multitask learning is particularly difficult, as it requires hyper-parameter selection for several different, but related tasks in a dependent manner. For the described approach, each

---

**Algorithm 1** Bundle Methods for Structured Output Algorithm

---
1: $S \geq 1$: maximal size of the bundle set
2: $\theta > 0$: linesearch trade-off (cf. [9] for details)
3: $\mathbf{w}^{(1)} = \mathbf{w}_p$
4: $k = 1$ and $\bar{\boldsymbol{a}} = \mathbf{0}, \bar{b} = 0, \Gamma_i = \emptyset \quad \forall i$
5: **repeat**
6:    **for** $i = 1, .., n$ **do**
7:       $\boldsymbol{y}^* = \text{argmax}_{\boldsymbol{y} \in \Upsilon} \{\langle \mathbf{w}^{(k)}, \Psi(\boldsymbol{x}_i, \boldsymbol{y}) \rangle + \Delta(\boldsymbol{y}_i, \boldsymbol{y})\}$
8:       **if** $\ell \left( \max\limits_{\boldsymbol{y} \in \Upsilon} \{\langle \mathbf{w}, \Psi(\boldsymbol{x}_i, \boldsymbol{y}) \rangle + \Delta(\boldsymbol{y}_i, \boldsymbol{y})\} \right) > \ell \left( \max\limits_{(\Psi, \Delta) \in \Gamma_i} \langle \mathbf{w}, \Psi \rangle + \Delta \right)$ **then**
9:         $\Gamma_i = \Gamma_i \cup (\Psi(\boldsymbol{x}_i, \boldsymbol{y}^*), \Delta(\boldsymbol{y}_i, \boldsymbol{y}^*))$
10:       **end if**
11:       Compute $\boldsymbol{a}_k \in \partial_{\mathbf{w}} \hat{L}(\mathbf{w}^{(k)})$
12:       Compute $b_k = \hat{L}(\mathbf{w}^{(k)}) - \langle \mathbf{w}^{(k)}, \boldsymbol{a}_k \rangle$
13:       $\mathbf{w}^* = \underset{\mathbf{w} \in \mathcal{H}}{\text{argmin}} \left\{ R_{p,\gamma}(\mathbf{w}) + \max \left( \max\limits_{(k-S)_+ < i \leq k} \{\langle \boldsymbol{a}_i, \mathbf{w} \rangle + b_i\}, \langle \bar{\boldsymbol{a}}, \mathbf{w} \rangle + \bar{b} \right) \right\}$
14:       Update $\bar{\boldsymbol{a}}, \bar{b}$ according to (6)
15:       $\eta^* = \text{argmin}_{\eta \in \Re} \hat{J}(\mathbf{w}^* + \eta(\mathbf{w}^* - \mathbf{w}^{(k)}))$
16:       $\mathbf{w}^{(k+1)} = (1 - \theta)\mathbf{w}^* + \theta\eta^*(\mathbf{w}^* - \mathbf{w}^{(k)})$
17:       $k = k + 1$
18:    **end for**
19: **until** $L(\mathbf{w}^{(k)}) - \hat{L}(\mathbf{w}^{(k)}) \leq \epsilon$ and $J(\mathbf{w}^{(k)}) - J_k(\mathbf{w}^{(k)}) \leq \epsilon$

---

node $n$ in the given taxonomy corresponds to solving an optimization problem that is subject to hyper-parameters $\gamma_n$ and $C_n$ (except for the root node, where only $C_n$ is relevant). Hence, the direct optimization of all combinations of dependent hyper-parameters in model selection is not feasible in many cases. Therefore, we propose to perform a *local* model selection and optimize the current $C_n$ and $\gamma_n$ at each node $n$ from top to bottom independently. This corresponds to using the taxonomy for reducing the parameter search space. To clarify this point, assume a perfect binary tree for $n$ tasks. The length of the path from root to leaf is $log_2(n)$. The parameters along one path are dependent, e.g. the values chosen at the root will influence the optimal choice further down the tree. Given $k$ candidate values for parameter $\gamma_n$, jointly optimizing all interdependent parameters along one path corresponds to optimizing over a grid of $k^{log_2(n)}$ in contrast to $k \cdot log_2(n)$ when using our proposed *local* strategy.

## 3 Results

### 3.1 Background

To demonstrate the validity of our approach, we applied it to the computational biology problem of gene finding. Here, the task is to identify genomic regions encoding genes (from which RNAs and/or proteins are produced). Genomic sequence can be represented by long strings of the four letters A, C, G, and T (genome sizes range from a few megabases to several gigabases). In prokaryotes (mostly bacteria and archaea) gene structures are comparably simple (cf. Figure 1A): the protein coding region starts by a start codon (one out of three specific 3-mers in many prokaryotes) followed by a number of codon triplets (of three nucleotides each) and is terminated by a stop codon (one out of five specific 3-mers in many prokaryotes). Genic regions are first transcribed to RNA, subsequently the contained coding region is translated into a protein. Parts of the RNA that are not translated are called untranslated region (UTR). Genes are separated from one another by intergenic regions. The protein coding segment is depleted of stop codons making the computational problem of identifying coding regions relatively straight forward.

In higher eukaryotes (animals, plants, etc.) however, the coding region can be interrupted by introns, which are removed from the RNA before it is translated into protein. Introns are flanked by specific sequence signals, so-called splice sites (cf. Figure 1B). The presence of introns substantially complicates the identification of the transcribed and coding regions. In particular, it is usually insufficient to identify regions depleted of stop codons to determine the encoded protein sequence. To

accurately detect the transcribed regions in eukaryotic genomes, it is therefore often necessary to use additional experimental data (e.g., sequencing of RNA fragments). Here, we consider two key problems in computational gene finding of (i) predicting (only) the coding regions for prokaryotes and (ii) predicting the exon-intron structure (but not the coding region) for eukaryotes.

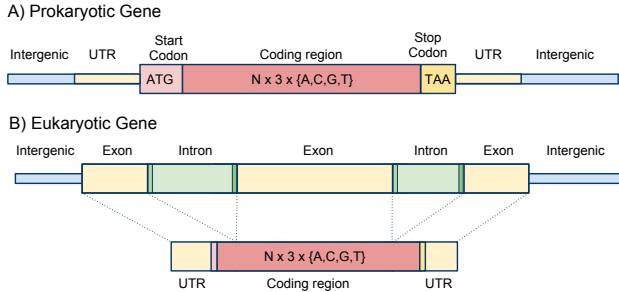

Figure 1: Panel A shows the structure of a prokaryotic gene. The protein coding region is flanked by a start and a stop codon and contains a multiple of three nucleotides. UTR denotes the untranslated region. Panel B shows the structure of an eukaryotic gene. The transcribed region contains introns and exons. Introns are flanked by splice sites and are removed from the RNA. The remaining sequence contains the UTRs and coding region.

The problem of identifying genes can be posed as a label sequence learning task, were one assigns a label (out of *intergenic, transcript start, untranslated region, coding start, coding exon, intron, coding stop, transcript stop*) to each position in the genome. The labels have to follow a grammar dictated by the biological processes of transcription and translation (see Figure 1) making it suitable to apply structured output learning techniques to identify genes. Because the biological processes and cellular machineries which recognize genes have slowly evolved over time, genes of closely related species tend to exhibit similar sequence characteristics. Therefore these problems are very well suited for the application of multitask learning: sharing information among species is expected to lead to more accurate gene predictions compared to approaching the problem for each species in isolation. Currently, the genomes of many prokaryotic and eukaryotic species are being sequenced, but often very little is known about the genes encoded, and standard methods are typically used to infer them without systematically exploiting reliable information on related species.

In the following we will consider two different aspects of the described problem. First, focusing on eukaryotic gene finding for a single species, we show that the proposed optimization algorithm very quickly converges to the optimal solution. Second, for the problem of prokaryotic gene finding in several species, we demonstrate that hierarchical multitask structured output learning significantly improves gene prediction accuracy. The supplement, data and code can be found on the project website[3].

## 3.2 Eukaryotic Gene Finding Based on RNA-Seq

We first consider the problem of detecting exonic, intronic and intergenic regions in a single eukaryotic genome. We use experimental data from RNA sequencing (RNA-seq) which provides evidence for exonic and intronic regions . For simplicity, we assume that for each position in the genome we are given numbers on how often this position was experimentally determined to be exonic and intronic, respectively. Ideally, exons and introns belonging to the same gene should have a constant number of confirmations, whereas these values may vary greatly between different genes. But in reality, these measurements are typically incomplete and noisy, so that inference techniques greatly help to reconstruct complete gene structures.

As any HMM or HMSVM, our method employs a state model defining allowed transitions between states. It consists of five basic states: *intergenic*, *exonic*, *intron start* (donor splice site), *intronic*, and *intron end* (acceptor splice site). These states are duplicated $Q = 5$ times to model different levels of confirmation and the whole model is mirrored for simultaneous predictions of genes from both strands of the genome (see supplement for details). In total, we have 41 states, each of which is associated with several parameters scoring features derived from the exon and intron confirmation and computational splice site predictions (see supplement for details). Overall the model has almost 1000 parameters.

We trained the model using 700 training regions with known exon/intron structures and a total length of ca. 5.2 million nucleotides (data from the nematode *C. elegans*). We used the column generation-based algorithm (see Appendix) and the Bundle method-based algorithm (Algorithm 1) and recorded upper and lower bounds of the objective during run time (cf. Figure 2). Whereas both algorithms

need a similar amount of computation per iteration (mostly decoding steps), the Bundle-method showed much faster convergence.

We assessed prediction accuracy in a three-fold cross-validation procedure where individual test sequences consisted of large genomic regions (of several Mbp) each containing many genes. This evaluation procedure is expected to yield unbiased estimates that are very similar to whole-genome predictions. Prediction accuracy was compared to another recently proposed, widely used method called Cufflinks [20]. We observed that our method detects introns and transcripts more accurately than Cufflinks in the data set analyzed here (cf. Figure 2).

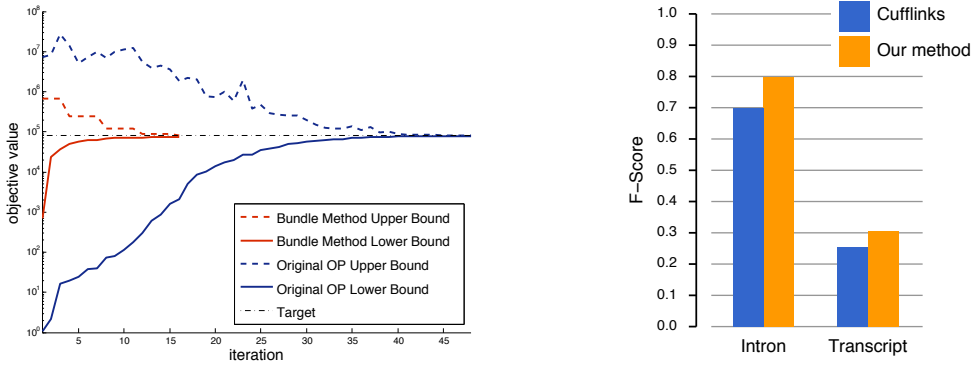

Figure 2: Left panel: Convergence for bundle method-based solver versus column generation (log-scale). Right panel: Prediction accuracy of our eukaryotic gene finding method in comparison to a state-of-the-art method, Cufflinks [20]. The F-score (harmonic mean of precision and recall) was assessed based on two metrics: correctly predicted introns as well as transcripts for which all introns were correct (see label).

### 3.3  Gene Finding in Multiple Prokaryotic Genomes

In a second series of experiments we evaluated the benefit of applying SO-MTL to prokaryotic gene prediction.

**SO prediction method**  We modeled prokaryotic genes as a Markov chain on the nucleotide level. To nonetheless account for the biological fact that genetic information is encoded in triplets, the model contains a 3-cycle of exon states; details are given in Figure 3.

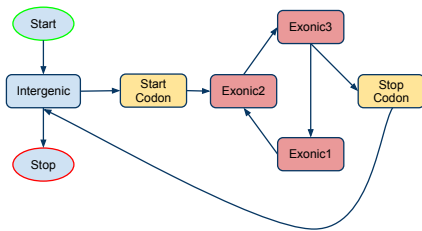

Figure 3: Simple state model for prokaryotic gene finding. A suitable model for prokaryotic gene prediction needs to consider 1) that a gene starts with a start codon (i.e. a certain triplet of nucleotides) 2) ends with a stop codon and 3) has a length divisible by 3. Properties 1) and 2) are enforced by allowing only transitions into and out of the exonic states on start and stop codons, respectively. Property 3) is enforced by only allowing transitions from exon state *Exonic3* to the stop codon state.

**Data generation**  We selected a subset of organisms with publicly available genomes to broadly cover the spectrum of prokaryotic organisms. In order to show that MTL is beneficial even for relatively distant species, we selected representatives from two different domains: bacteria and archaea. The relationship between these organisms is captured by the taxonomy shown in Figure 4, which was created based on the information available on the NCBI website[4]. For each organism, we generated one training example per annotated gene. The genomic sequences were cut between neighboring genes (splitting intergenic regions equally), such that a minimum distance of 6 nucleotides between genes was maintained. Features for SO learning were derived from the nucleotide sequence by transcoding it to a numerical representation of triplets. This resulted in binary vectors of size $4^3 = 64$ with exactly one non-zero entry. We sub-sampled from the complete dataset of $N_i$ examples for each organism $i$ and created new datasets with 20 training examples, 40 evaluation examples and 200 test examples.

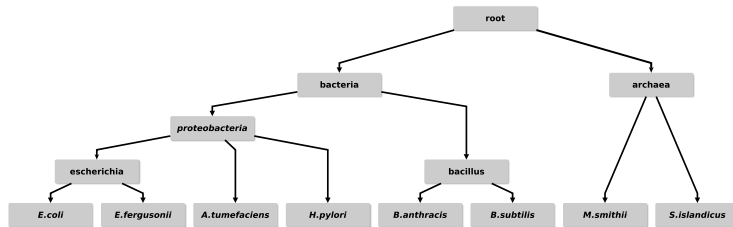

Figure 4: Species and their taxonomic hierarchy used for prokaryotic gene finding.

**Experimental setup**   For model selection we used a grid over the following two parameter ranges $C = [100, 250], \gamma = [0, 0.025, 0.1, 0.25, 0.5, 0.75, 0.9, 1.0]$ for each node in the taxonomy (cf. Figure 4). Sub-sampling of the dataset was performed 3 times and results were subsequently averaged. We compared our MTL algorithm to two baseline methods, one where predictors for all tasks where trained without information transfer (*independent*) and the other extreme case, where one global model was fitted for all tasks based on the union of all data sets (*union*). Performance was measured by the F-score, the harmonic mean of precision and recall, where precision and recall were determined on nucleotide level (e.g. whether or not an exonic nucleotide was correctly predicted) in single-gene regions. (Note that due to its per-nucleotide Markov restriction, however, our method is not able to exploit that there is only one gene per examples sequence.)

**Results**   Figure 5 shows the results for our proposed MTL method and the two baseline methods described above (see Appendix for table). We observe that it generally pays off to combine information from different organisms, as *union* always performs better than *independent*. Indeed MTL improves over the naive combination method *union* with F-score increases of up to $4.05$ percentage points in *A. tumefaciens*. On average, we observe an improvement of $13.99$ percentage points for MTL over *independent* and $1.13$ percentage points for MTL over *union*, confirming the value of MTL in transferring information across tasks. In addition, the new bundle method converges at least twice as fast as the originally proposed cutting plane method.

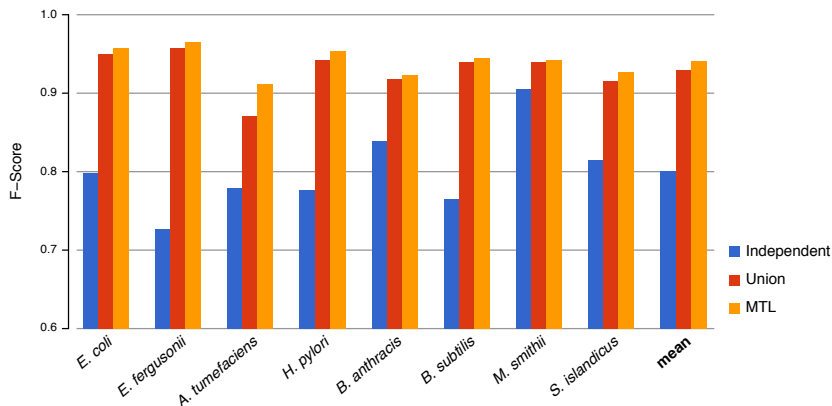

Figure 5: Evaluation of MTL and baseline methods *independent* and *union*.

## 4   Discussion

We have introduced a regularization-based approach to SO learning in the setting of hierarchical task relations and have empirically shown its validity on an application from computational biology. To cope with the increased problem size usually encountered in the MTL setting, we have developed an efficient solver based on bundle-methods and demonstrated its improved convergence behavior compared to column generation techniques. Applying our SO-MTL algorithm to the problem of prokaryotic gene finding, we could show that sharing information across tasks indeed results in improved accuracy over learning tasks in isolation. Additionally, the taxonomy, which relates individual tasks to each other, proved useful in that it led to more accurate predictions than were obtained when simply training on all examples together. We have previously shown that MTL algorithms excel in a scenarios where there is limited training data relative to the complexity of the problem and model [23]. As this experiment was carried out on a relatively small data set, more work is required to turn our approach into a state-of-the-art prokaryotic gene finder.

**Acknowledgments**

We would like to thank the anonymous reviewers for insightful comments. Moreover, we are grateful to Jonas Behr, Jose Leiva, Yasemin Altun and Klaus-Robert Müller. This work was supported by the German Research Foundation (DFG) under the grant RA 1894/1-1.

## Footnotes

[1]These authors contributed equally.

[2]This work was done while SS was at Technical University Berlin

[3]http://bioweb.me/so-mtl

[4] ftp://ftp.ncbi.nlm.nih.gov/genomes/Bacteria/

# References

[1] A. Agarwal, S. Gerber, and H. Daumé III. Learning multiple tasks using manifold regularization. In *Advances in Neural Information Processing Systems 23*, 2010.

[2] Y. Altun, I. Tsochantaridis, and T. Hofmann. Hidden markov support vector machines. In *Proc. ICML*, 2003.

[3] S. Ben-David and R. Schuller. Exploiting task relatedness for multiple task learning. *Lecture notes in computer science*, pages 567–580, 2003.

[4] J. Blitzer, K. Crammer, A. Kulesza, F. Pereira, and J. Wortman. Learning bounds for domain adaptation. *Advances in Neural Information Processing Systems*, 20, 2007.

[5] R. Caruana. Multitask learning. *Machine Learning*, 28(1):41–75, 1997.

[6] H. Daumé III. Bayesian multitask learning with latent hierarchies. In *Proceedings of the Twenty-Fifth Conference on Uncertainty in Artificial Intelligence*, 2009.

[7] T.-M.-T. Do. *Regularized Bundle Methods for Large-scale Learning Problems with an Application to Large Margin Training of Hidden Markov Models*. PhD thesis, l'Université Pierre & Marie Curie, 2010.

[8] T. Evgeniou, C. Micchelli, and M. Pontil. Learning multiple tasks with kernel methods. *Journal of Machine Learning Research*, 6:615–637, 2005.

[9] V. Franc and S. Sonnenburg. OCAS optimized cutting plane algorithm for support vector machines. In *Proc. ICML*, 2008.

[10] T. Hazan and R. Urtasun. A primal-dual message-passing algorithm for approximated large scale structured prediction. In *Advances in Neural Information Processing Systems 23*, 2010.

[11] L. Jacob, F. Bach, and J. Vert. Clustered multi-task learning: A convex formulation. *Arxiv preprint arXiv:0809.2085*, 2008.

[12] L. Jacob and J. Vert. Efficient peptide-MHC-I binding prediction for alleles with few known binders. *Bioinformatics*, 24(3):358–66, 2008.

[13] S. Kim and E. P. Xing. Tree-guided group lasso for multi-task regression with structured sparsity. *Proc. ICML*, 2010.

[14] J. Lafferty, A. McCallum, and F. Pereira. Conditional random fields: Probabilistic models for segmenting and labeling sequence data. In *Proc. ICML*, 2001.

[15] G. Rätsch and S. Sonnenburg. Large scale hidden semi-markov SVMs. In *Advances in Neural Information Processing Systems 18*, 2006.

[16] G. Schweikert, C. Widmer, B. Schölkopf, and G. Rätsch. An Empirical Analysis of Domain Adaptation Algorithms for Genomic Sequence Analysis. In *Advances in Neural Information Processing Systems 21*, 2009.

[17] G. Schweikert, A. Zien, G. Zeller, J. Behr, C. Dieterich, C. Ong, P. Philips, F. De Bona, L. Hartmann, A. Bohlen, N. Krüger, S. Sonnenburg, and G. Rätsch. mGene: accurate SVM-based gene finding with an application to nematode genomes. *Genome Research*, 19(11):2133–43, 2009.

[18] A. Smola, S. Vishwanathan, and Q. Le. Bundle methods for machine learning. In *Advances in Neural Information Processing Systems 20*, 2008.

[19] C. Teo, S. Vishwanathan, A.Smola, and Q. Le. Bundle methods for regularized risk minimization. *Journal of Machine Learning Research*, 11:311–365, 2010.

[20] C. Trapnell, B. A. Williams, G. Pertea, A. Mortazavi, G. Kwan, M. J. van Baren, S. L. Salzberg, B. J. Wold, and L. Pachter. Transcript assembly and quantification by RNA-seq reveals unannotated transcripts and isoform switching during cell differentiation. *Nature Biotechnology*, 28:511–515, 2010.

[21] I. Tsochantaridis, T. Joachims, T. Hofmann, and Y. Altun. Large margin methods for structured and interdependent output variables. *Journal of Machine Learning Research*, 6:1453–1484, 2005.

[22] C. Widmer, J. Leiva, Y. Altun, and G. Rätsch. Leveraging Sequence Classification by Taxonomy-based Multitask Learning. In *Research in Computational Molecular Biology*, 2010.

[23] C. Widmer, N. Toussaint, Y. Altun, and G. Rätsch. Inferring latent task structure for Multitask Learning by Multiple Kernel Learning. *BMC Bioinformatics*, 11(Suppl 8):S5, 2010.

